# Accelerated Mini-batch Randomized Block Coordinate Descent Method

**Tuo Zhao**[†§*]    **Mo Yu**[‡*]  **Yiming Wang**[†]   **Raman Arora**[†]   **Han Liu**[§]
[†]Johns Hopkins University   [‡]Harbin Institute of Technology   [§]Princeton University
{tour,myu25,freewym,arora}@jhu.edu,hanliu@princeton.edu

## Abstract

We consider regularized empirical risk minimization problems. In particular, we minimize the sum of a smooth empirical risk function and a nonsmooth regularization function. When the regularization function is block separable, we can solve the minimization problems in a randomized block coordinate descent (RBCD) manner. Existing RBCD methods usually decrease the objective value by exploiting the partial gradient of a randomly selected block of coordinates in each iteration. Thus they need all data to be accessible so that the partial gradient of the block gradient can be exactly obtained. However, such a "batch" setting may be computationally expensive in practice. In this paper, we propose a mini-batch randomized block coordinate descent (MRBCD) method, which estimates the partial gradient of the selected block based on a mini-batch of randomly sampled data in each iteration. We further accelerate the MRBCD method by exploiting the semi-stochastic optimization scheme, which effectively reduces the variance of the partial gradient estimators. Theoretically, we show that for strongly convex functions, the MRBCD method attains lower overall iteration complexity than existing RBCD methods. As an application, we further trim the MRBCD method to solve the regularized sparse learning problems. Our numerical experiments shows that the MRBCD method naturally exploits the sparsity structure and achieves better computational performance than existing methods.

## 1  Introduction

Big data analysis challenges both statistics and computation. In the past decade, researchers have developed a large family of sparse regularized M-estimators, such as Sparse Linear Regression [17, 24], Group Sparse Linear Regression [22], Sparse Logistic Regression [9], Sparse Support Vector Machine [23, 19], and etc. These estimators are usually formulated as regularized empirical risk minimization problems in a generic form as follows [10],

$$\widehat{\boldsymbol{\theta}} = \underset{\boldsymbol{\theta}}{\operatorname{argmin}} \, \mathcal{P}(\boldsymbol{\theta}) = \underset{\boldsymbol{\theta}}{\operatorname{argmin}} \, \mathcal{F}(\boldsymbol{\theta}) + \mathcal{R}(\boldsymbol{\theta}), \tag{1.1}$$

where $\boldsymbol{\theta}$ is the parameter of the working model. Here we assume the empirical risk function $\mathcal{F}(\boldsymbol{\theta})$ is smooth, and the regularization function $\mathcal{R}(\boldsymbol{\theta})$ is non-differentiable. Some first order algorithms, mostly variants of proximal gradient methods [11], have been proposed for solving (1.1) . For strongly convex $\mathcal{P}(\boldsymbol{\theta})$, these methods achieve linear rates of convergence [1].

The proximal gradient methods, though simple, are not necessarily efficient for large problems. Note that empirical risk function $\mathcal{F}(\boldsymbol{\theta})$ is usually composed of many smooth component functions:

$$\mathcal{F}(\boldsymbol{\theta}) = \frac{1}{n} \sum_{i=1}^{n} f_i(\boldsymbol{\theta}) \quad \text{and} \quad \nabla \mathcal{F}(\boldsymbol{\theta}) = \frac{1}{n} \sum_{i=1}^{n} \nabla f_i(\boldsymbol{\theta}),$$

---

[*]Both authors contributed equally.

where each $f_i$ is associated with a few samples of the whole date set. Since the proximal gradient methods need to calculate the gradient of $\mathcal{F}$ in every iteration, the computational complexity scales linearly with the sample size (or the number of components functions). Thus the overall computation can be expensive especially when the sample size is very large in such a "batch" setting [16].

To overcome the above drawback, recent work has focused on stochastic proximal gradient methods (SPG), which exploit the additive nature of the empirical risk function $\mathcal{F}(\boldsymbol{\theta})$. In particular, the SPG methods randomly sample only a few $f_i$'s to estimate the gradient $\nabla \mathcal{F}(\boldsymbol{\theta})$, i.e., given an index set $\mathcal{B}$, also as known as a mini-batch [16], where all elements are independently sampled from $\{1, ..., n\}$ with replacement, we consider a gradient estimator $\frac{1}{|\mathcal{B}|} \sum_{i \in \mathcal{B}} \nabla f_i(\boldsymbol{\theta})$. Thus calculating such a "stochastic" gradient can be far less expensive than the proximal gradient methods within each iteration. Existing literature has established the global convergence results for the stochastic proximal gradient methods [3, 7] based on the unbiasedness of the gradient estimator, i.e.,

$$\mathbb{E}_{\mathcal{B}} \left[ \frac{1}{|\mathcal{B}|} \sum_{i \in \mathcal{B}} \nabla f_i(\boldsymbol{\theta}) \right] = \nabla \mathcal{F}(\boldsymbol{\theta}) \quad \text{for} \ \forall \ \boldsymbol{\theta} \in \mathbb{R}^d.$$

However, owing to the variance of the gradient estimator introduced by the stochastic sampling, SPG methods only achieve sublinear rates of convergence even when $\mathcal{P}(\boldsymbol{\theta})$ is strongly convex [3, 7].

A second line of research has focused randomized block coordinate descent (RBCD) methods. These methods exploit the block separability of the regularization function $\mathcal{R}$, i.e., given a partition $\{\mathcal{G}_1, ..., \mathcal{G}_k\}$ of $d$ coordinates, we use $\boldsymbol{v}_{\mathcal{G}_j}$ to denote the subvector of $\boldsymbol{v}$ with all indices in $\mathcal{G}_j$, and then we can write

$$\mathcal{R}(\boldsymbol{\theta}) = \sum_{j=1}^{k} r_j(\boldsymbol{\theta}_{\mathcal{G}_j}) \quad \text{with} \ \boldsymbol{\theta} = (\boldsymbol{\theta}_{\mathcal{G}_1}^T, ..., \boldsymbol{\theta}_{\mathcal{G}_k}^T)^T.$$

Accordingly, they develop the randomized block coordinate descent (RBCD) methods. In particular, the block coordinate descent methods randomly select a block of coordinates in each iteration, and then only calculate the gradient of $\mathcal{F}$ with respect to the selected block [15, 13]. Since the variance introduced by the block selection asymptotically goes to zero, the RBCD methods also attain linear rates of convergence when $\mathcal{P}(\boldsymbol{\theta})$ is strongly convex. For sparse learning problems, the RBCD methods have a natural advantage over the proximal gradient methods. Because many blocks of coordinates stay at zero values throughout most of iterations, we can integrate the active set strategy into the computation. The active set strategy maintains an only iterates over a small subset of all blocks [2], which greatly boosts the computational performance. Recent work has corroborated the empirical advantage of RBCD methods over the proximal gradient method [4, 20, 8]. The RBCD methods, however, still requires that all component functions are accessible within every iteration so that the partial gradient can be exactly obtained.

To address this issue, we propose a stochastic variant of the RBCD methods, which shares the advantage with both the SPG and RBCD methods. More specifically, we randomly select a block of coordinates in each iteration, and estimate the corresponding partial gradient based on a mini-batch of $f_i$'s sampled from all component functions. To address the variance introduced by stochastic sampling, we exploit the semi-stochastic optimization scheme proposed in [5, 6]. The semi-stochastic optimization scheme contains two nested loops: For each iteration of the outer loop, we calculate an exact gradient. Then in the follow-up inner loop, we adjust all estimated partial gradients by the obtained exact gradient. Such a modification, though simple, has a profound impact: the amortized computational complexity in each iteration is similar to the stochastic optimization, but the rate of convergence is not compromised. Theoretically, we show that when $\mathcal{P}(\boldsymbol{\theta})$ is strongly convex, the MRBCD method attains better overall iteration complexity than existing RBCD methods. We then apply the MRBCD method combined with the active set strategy to solve the regularized sparse learning problems. Our numerical experiments shows that the MRBCD method achieves much better computational performance than existing methods.

A closely related method is the stochastic proximal variance reduced gradient method proposed in [21]. Their method is a variant of the stochastic proximal gradient methods using the same semi-stochastic optimization scheme as ours, but their method inherits the same drawback as the proximal gradient method, and does not fully exploit the underlying sparsity structure for large sparse learning problems. We will compare its computational performance with the MRBCD method in numerical

experiments. Note that their method can be viewed as a special example of the MRBCD method with one single block.

While this paper was under review, we learnt that a similar method was independently proposed by [18]. They also apply the variance reduction technique into the randomized block coordinate descent method, and obtain similar theoretical results to ours.

## 2 Notations and Assumptions

Given a vector $\boldsymbol{v} = (v_1, ..., v_d)^T \in \mathbb{R}^d$, we define vector norms: $||\boldsymbol{v}||_1 = \sum_j |v_j|$, $||\boldsymbol{v}||^2 = \sum_j v_j^2$, and $||\boldsymbol{v}||_\infty = \max_j |v_j|$. Let $\{\mathcal{G}_1, ..., \mathcal{G}_k\}$ be a partition of all $d$ coordinates with $|\mathcal{G}_j| = p_j$ and $\sum_{j=1}^k p_j = d$. We use $\boldsymbol{v}_{\mathcal{G}_j}$ to denote the subvector of $\boldsymbol{v}$ with all indices in $\mathcal{G}_j$, and $\boldsymbol{v}_{\backslash \mathcal{G}_j}$ to denote the subvector of $\boldsymbol{v}$ with all indices in $\mathcal{G}_j$ removed.

Throughout the rest of the paper, if not specified, we make the following assumptions on $\mathcal{P}(\boldsymbol{\theta})$.

**Assumption 2.1.** *Each $f_i(\boldsymbol{\theta})$ is convex and differentiable. Given the partition $\{\mathcal{G}_1, ..., \mathcal{G}_k\}$, all $\nabla_{\mathcal{G}_j} f_i(\boldsymbol{\theta}) = [\nabla f_i(\boldsymbol{\theta})]_{\mathcal{G}_j}$'s are Lipschitz continuous, i.e., there exists a positive constants $L_{\max}$ such that for all $\boldsymbol{\theta}, \boldsymbol{\theta}' \in \mathbb{R}^d$ and $\boldsymbol{\theta}_{\mathcal{G}_j} \neq \boldsymbol{\theta}'_{\mathcal{G}_j}$, we have*

$$||\nabla_{\mathcal{G}_j} f_i(\boldsymbol{\theta}) - \nabla_{\mathcal{G}_j} f_i(\boldsymbol{\theta}')|| \leq L_{\max} ||\boldsymbol{\theta}_{\mathcal{G}_j} - \boldsymbol{\theta}'_{\mathcal{G}_j}||.$$

*Moreover, $\nabla f_i(\boldsymbol{\theta})$ is Lipschitz continuous, i.e., there exists a positive constant $T_{\max}$ for all $\boldsymbol{\theta}, \boldsymbol{\theta}' \in \mathbb{R}^d$ and $\boldsymbol{\theta} \neq \boldsymbol{\theta}'$, we have*

$$||\nabla f_i(\boldsymbol{\theta}) - \nabla f_i(\boldsymbol{\theta}')|| \leq T_{\max} ||\boldsymbol{\theta} - \boldsymbol{\theta}'||.$$

Assumption 2.1 also implies that $\nabla \mathcal{F}(\boldsymbol{\theta})$ is Lipschitz continuous, and given the tightest $T_{\max}$ and $L_{\max}$ in Assumption 2.1, we have $T_{\max} \leq kL_{\max}$.

**Assumption 2.2.** *$F(\boldsymbol{\theta})$ is strongly convex, i.e., for all $\boldsymbol{\theta}$ and $\boldsymbol{\theta}'$, there exists a positive constant $\mu$ such that*

$$\mathcal{F}(\boldsymbol{\theta}') - \mathcal{F}(\boldsymbol{\theta}) + \nabla \mathcal{F}(\boldsymbol{\theta})^T(\boldsymbol{\theta}' - \boldsymbol{\theta}) \geq \frac{\mu}{2} ||\boldsymbol{\theta}' - \boldsymbol{\theta}||^2.$$

Note that Assumption 2.2 also implies that $\mathcal{P}(\boldsymbol{\theta})$ is strongly convex.

**Assumption 2.3.** *$\mathcal{R}(\boldsymbol{\theta})$ is a simple convex nonsmooth function such that given some positive constant $\eta$, we can obtain a closed form solution to the following optimization problem,*

$$\mathcal{T}_\eta^j(\boldsymbol{\theta}'_{\mathcal{G}_j}) = \underset{\boldsymbol{\theta}_{\mathcal{G}_j} \in \mathbb{R}^{p_j}}{\operatorname{argmin}} \frac{1}{2\eta} ||\boldsymbol{\theta}_{\mathcal{G}_j} - \boldsymbol{\theta}'_{\mathcal{G}_j}||^2 + r_j(\boldsymbol{\theta}).$$

Assumptions 2.1-2.3 are satisfied by many popular regularized empirical risk minimization problems. We give some examples in the experiments section.

## 3 Method

The MRBCD method is doubly stochastic, in the sense that we not only randomly select a block of coordinates, but also randomly sample a mini-batch of component functions from all $f_i$'s. The partial gradient of the selected block is estimated based on the selected component functions, which yields a much lower computational complexity than existing RBCD methods in each iteration.

A naive implementation of the MRBCD method is summarized in Algorithm 1. Since the variance introduced by stochastic sampling over component functions does not go to zero as the number of iteration increases, we have to choose a sequence of diminishing step sizes (e.g. $\eta_t = \mu^{-1} t^{-1}$) to ensure the convergence. When $t$ is large, we only gain very limited descent in each iteration. Thus the MRBCD-I method can only attain a sublinear rate of convergence.

---

**Algorithm 1** Mini-batch Randomized Block Coordinate Descent Method-I: A Naive Implementation. The stochastic sampling over component functions introduces variance to the partial gradient estimator. To ensure the convergence, we adopt a sequence of diminishing step sizes, which eventually leads to sublinear rates of convergence.

---

**Parameter:** Step size $\eta_t$
**Initialize:** $\boldsymbol{\theta}^{(0)}$
**For** $t = 1, 2, ...$
    Randomly sample a mini-batch $\mathcal{B}$ from $\{1, ..., n\}$ with equal probability
    Randomly sample $j$ from $\{1, ..., k\}$ with equal probability
    $\boldsymbol{\theta}_{\mathcal{G}_j}^{(t)} \leftarrow \mathcal{T}_{\eta_t}^j \left( \boldsymbol{\theta}_{\mathcal{G}_j}^{(t-1)} - \eta_t \nabla_{\mathcal{G}_j} f_{\mathcal{B}}(\boldsymbol{\theta}^{(t-1)}) \right), \boldsymbol{\theta}_{\backslash G_j}^{(t)} \leftarrow \boldsymbol{\theta}_{\backslash G_j}^{(t-1)}$
**End for**

---

## 3.1 MRBCD with Variance Reduction

A recent line of work shows how to reduce the variance in the gradient estimation without deteriorating rates of convergence using a semi-stochastic optimization scheme [5, 6]. The semi-stochastic optimization contains two nested loops: In each iteration of the outer loop, we calculate an exact gradient; Then within the follow-up inner loop, we use the obtained exact gradient to adjust all estimated partial gradients. These adjustments can guarantee that the variance introduced by stochastic sampling over component functions asymptotically goes to zero (see [5]).

---

**Algorithm 2** Mini-batch Randomized Block Coordinate Descent Method-II: MRBCD + Variance Reduction. We periodically calculate the exact gradient at the beginning of each outer loop, and then use the obtained exact gradient to adjust all follow-up estimated partial gradients. These adjustments guarantee that the variance introduced by stochastic sampling over component functions asymptotically goes to zero, and help the MRBCD II method attain linear rates of convergence.

---

**Parameter:** update frequency $m$ and step size $\eta$
**Initialize:** $\widetilde{\boldsymbol{\theta}}^{(0)}$
**For** s = 1,2,...
    $\widetilde{\boldsymbol{\theta}} \leftarrow \widetilde{\boldsymbol{\theta}}^{(s-1)}, \widetilde{\boldsymbol{\mu}} \leftarrow \nabla\mathcal{F}(\widetilde{\boldsymbol{\theta}}^{(s-1)}), \boldsymbol{\theta}^{(0)} \leftarrow \widetilde{\boldsymbol{\theta}}^{(s-1)}$
    **For** $t = 1, 2, ..., m$
    Randomly sample a mini-batch $\mathcal{B}$ from $\{1, ..., n\}$ with equal probability
    Randomly sample $j$ from $\{1, ..., k\}$ with equal probability
        $\boldsymbol{\theta}_{\mathcal{G}_j}^{(t)} \leftarrow \mathcal{T}_{\eta}^j \left( \boldsymbol{\theta}_{\mathcal{G}_j}^{(t-1)} - \eta \left[ \nabla_{\mathcal{G}_j} f_{\mathcal{B}}(\boldsymbol{\theta}^{(t-1)}) - \nabla_{\mathcal{G}_j} f_{\mathcal{B}}(\widetilde{\boldsymbol{\theta}}) + \widetilde{\boldsymbol{\mu}}_{\mathcal{G}_j} \right] \right), \boldsymbol{\theta}_{\backslash G_j}^{(t)} \leftarrow \boldsymbol{\theta}_{\backslash G_j}^{(t-1)}$
    **End for**
    $\widetilde{\boldsymbol{\theta}}^{(s)} \leftarrow \sum_{l=1}^{m} \boldsymbol{\theta}^{(l)}$
**End for**

---

The MRBCD method with variance reduction is summarized in Algorithm 2. In the next section, we will show that the MRBCD II method attains linear rates of convergence, and the amortized computational complexity within each iteration is almost the same as that of the MRBCD I method.

**Remark 3.1.** *Another option for the variance reduction is the stochastic averaging scheme as proposed in [14], which stores the gradients of most recently subsampled component functions. But the MRBCD method iterates randomly over different blocks of coordinates, which makes the stochastic averaging scheme inapplicable.*

## 3.2 MRBCD with Variance Reduction and Active Set Strategy

When applying the MRBCD II method to regularized sparse learning problems, we further incorporate the active set strategy to boost the empirical performance. Different from existing RBCD methods, which usually identify the active set by cyclic search, we exploit a proximal gradient pilot to identify the active set. More specifically, within each iteration of the outer loop, we conduct a proximal gradient descent step, and select the support of the resulting solution as the active set. This is very natural to the MRBCD-II method. Because at the beginning of each outer loop, we always calculate an exact gradient, and delivering a proximal gradient pilot will not introduce much addi-

tional computational cost. Once the active set is identified, all randomized block coordinate descent steps within the follow-up inner loop only iterates over blocks of coordinates in the active set.

---

**Algorithm 3** Mini-batch Randomized Block Coordinate Descent Method-III: MRBCD with Variance Reduction and Active Set. To fully take advantage of the obtained exact gradient, we adopt a proximal gradient pilot $\boldsymbol{\theta}^{(0)}$ to identify the active set at each iteration of the outer loop. Then all randomized coordinate descent steps within the follow-up inner loop only iterate over blocks of coordinates in the active set.

---

**Parameter:** update frequency $m$ and step size $\eta$
**Initialize:** $\widetilde{\boldsymbol{\theta}}^{(0)}$
**For** s = 1,2,...
$\quad \widetilde{\boldsymbol{\theta}} \leftarrow \widetilde{\boldsymbol{\theta}}^{(s-1)}, \widetilde{\boldsymbol{\mu}} \leftarrow \nabla \mathcal{F}(\widetilde{\boldsymbol{\theta}}^{(s-1)})$
$\quad$**For** $j = 1, 2, ..., k$
$\quad\quad \boldsymbol{\theta}_{\mathcal{G}_j}^{(0)} \leftarrow \mathcal{T}_{\eta/k}^j \left( \widetilde{\boldsymbol{\theta}}_{\mathcal{G}_j} - \eta \widetilde{\boldsymbol{\mu}}_{\mathcal{G}_j}/k \right)$
$\quad$**End for**
$\quad \mathcal{A} \leftarrow \{ j \mid \boldsymbol{\theta}_{\mathcal{G}_j}^{(0)} \neq \mathbf{0}\}, |\mathcal{B}| = |\mathcal{A}|$
$\quad$**For** $t = 1, 2, ..., m|\mathcal{A}|/k$
$\quad\quad$Randomly sample a mini-batch $\mathcal{B}$ from $\{1, ..., n\}$ with equal probability
$\quad\quad$Randomly sample $j$ from $\{1, ..., k\}$ with equal probability
$\quad\quad$**For all** $j \in \widetilde{\mathcal{A}}$
$\quad\quad \boldsymbol{\theta}_{\mathcal{G}_j}^{(t)} \leftarrow \mathcal{T}_{\eta}^j \left( \boldsymbol{\theta}_{\mathcal{G}_j}^{(t-1)} - \eta \left[ \nabla_{\mathcal{G}_j} f_{\mathcal{B}}(\boldsymbol{\theta}^{(t-1)}) - \nabla_{\mathcal{G}_j} f_{\mathcal{B}}(\widetilde{\boldsymbol{\theta}}) + \widetilde{\boldsymbol{\mu}}_{\mathcal{G}_j} \right] \right), \boldsymbol{\theta}_{\backslash G_j}^{(t)} \leftarrow \boldsymbol{\theta}_{\backslash G_j}^{(t-1)}$
$\quad$**End for**
$\quad \widetilde{\boldsymbol{\theta}}^{(s)} \leftarrow \sum_{l=1}^{m} \boldsymbol{\theta}^{(l)}$
**End for**

---

The MRBCD method with variance reduction and active set strategy is summarized in Algorithm 3. Since we integrate the active set into the computation, a successive $|\mathcal{A}|$ coordinate decent iterations in MRBCD-III will have similar performance as $k$ iterations in MRBCD-II. Therefore we change the maximum number of iterations within each inner loop to $|\mathcal{A}|m/k$. Moreover, since the support is only $|\mathcal{A}|$ blocks of coordinates, we only need to take $|\mathcal{B}| = |\mathcal{A}|$ to guarantee sufficient variance reduction. These modifications will further boost the computational performance of MRBCD-III.

**Remark 3.2.** *The exact gradient can be also used to determine the convergence of the MRBCD-III method. We terminate the iteration when the approximate KKT condition is satisfied, $\min_{\boldsymbol{\xi} \in \partial \mathcal{R}(\widetilde{\boldsymbol{\theta}})} ||\widetilde{\boldsymbol{\mu}} + \boldsymbol{\xi}|| \leq \varepsilon$, where $\varepsilon$ is a positive preset convergence parameter. Since evaluating whether the approximate KKT condition holds is based on the exact gradient obtained at each iteration of the outer loop, it does not introduce much additional computational cost, either.*

## 4 Theory

Before we proceed with our main results of the MRBCD-II method, we first introduce the important lemma for controlling the variance introduced by stochastic sampling.

**Lemma 4.1.** *Let $\mathcal{B}$ be a mini-batch sampled from $\{1, ..., n\}$. Define $\boldsymbol{v}_{\mathcal{B}} = \frac{1}{|\mathcal{B}|} \sum_{i \in |\mathcal{B}|} \nabla f_i(\boldsymbol{\theta}^{(t-1)}) - \frac{1}{|\mathcal{B}|} \sum_{i \in |\mathcal{B}|} \nabla f_i(\widetilde{\boldsymbol{\theta}}) + \widetilde{\boldsymbol{\mu}}$. Conditioning on $\boldsymbol{\theta}^{(t-1)}$, we have $\mathbb{E}_{\mathcal{B}} \boldsymbol{v}_{\mathcal{B}} = \nabla \mathcal{F}(\boldsymbol{\theta}^{(t-1)})$ and*

$$\mathbb{E}_{\mathcal{B}} ||\boldsymbol{v}_{\mathcal{B}} - \nabla \mathcal{F}(\boldsymbol{\theta}^{(t-1)})||^2 \leq \frac{4T_{max}}{|\mathcal{B}|} \left[ \mathcal{P}(\boldsymbol{\theta}^{(t-1)}) - \mathcal{P}(\widehat{\boldsymbol{\theta}}) + \mathcal{P}(\widetilde{\boldsymbol{\theta}}) - \mathcal{P}(\widehat{\boldsymbol{\theta}}) \right].$$

The proof of Lemma 4.1 is provided in Appendix A. Lemma 4.1 guarantees that $\boldsymbol{v}$ is an unbiased estimator of $\mathcal{F}(\boldsymbol{\theta})$, and its variance is bounded by the objective value gap. Therefore we do not need to choose a sequence diminishing step sizes to reduce the variance.

### 4.1 Strongly Convex Functions

We then present the concrete rates of convergence of MRBCD-II when $\mathcal{P}$ is strongly convex.

**Theorem 4.2.** *Suppose that Assumptions 2.1-2.3 hold. Let $\widetilde{\boldsymbol{\theta}}^{(s)}$ be a random point generated by the MRBCD-II method in Algorithm 2. Given a large enough batch $\mathcal{B}$ and a small enough learning rate $\eta$ such that $|\mathcal{B}| \geq T_{\max}/L_{\max}$ and $\eta < L_{\max}^{-1}/4$, we have*

$$\mathbb{E}\mathcal{P}(\widetilde{\boldsymbol{\theta}}^{(s)}) - \mathcal{P}(\widehat{\boldsymbol{\theta}}) \leq \left( \frac{k}{\mu\eta(1 - 4\eta L_{\max})m} + \frac{4\eta L_{\max}(m+1)}{(1 - 4\eta L_{\max})m} \right)^s [\mathcal{P}(\widetilde{\boldsymbol{\theta}}^{(0)}) - \mathcal{P}(\widehat{\boldsymbol{\theta}})].$$

Here we only present a sketch. The detailed proof is provided in Appendix B. The expected successive descent of the objective value is composed of two terms: The first one is the same as the expected successive descent of the "batch" RBCD methods; The second one is the variance introduced by the stochastic sampling. The descent term can be bounded by taking the average of the successive descent over all blocks of coordinates. The variance term can be bounded using Lemma 4.1. The mini-batch sampling and adjustments of $\boldsymbol{\mu}$'s guarantees that the variance asymptotically goes to zero at a proper scale. By taking expectation over the randomness of component functions and blocks of coordinates throughout all iterations, we derive a geometric rate of convergence.

The next corollary present the concrete iteration complexity of the MRBCD-II method.

**Corollary 4.3.** *Suppose that Assumptions 2.1-2.3 hold. Let $|\mathcal{B}| = T_{\max}/L_{\max}$, $m = 65kL_{\max}/\mu$, and $\eta = L_{\max}^{-1}/16$. Given the target accuracy $\epsilon$ and some $\rho \in (0,1)$, for any*

$$s \geq 3\log[\mathcal{P}(\widetilde{\boldsymbol{\theta}}^{(0)}) - \mathcal{P}(\widehat{\boldsymbol{\theta}})/\rho] + 3\log(1/\epsilon),$$

*we have $\mathcal{P}(\widetilde{\boldsymbol{\theta}}^{(s)}) - \mathcal{P}(\widehat{\boldsymbol{\theta}}) \leq \epsilon$ with at last probability $1 - \rho$.*

Corollary 4.3 is a direct result of Theorem 4.2 and Markov inequality. The detailed proof is provided in Appendix C.

To characterize the overall iteration complexity, we count the number of partial gradients we estimate. In each iteration of the outer loop, we calculate an exact gradient. Thus the number of estimated partial gradients is $\mathcal{O}(nk)$. Within each iteration of the inner loop ($m$ in total), we estimate the partial gradients based on a mini-batch $\mathcal{B}$. Thus the number of estimate partial gradients is $\mathcal{O}(m|\mathcal{B}|)$. If we choose $\eta$, $m$, and $\mathcal{B}$ as in Corollary (4.3) and consider $\rho$ as a constant, then the iteration complexity of the MRBCD-II method with respect to the number of estimated partial gradients is

$$\mathcal{O}\left((nk + kT_{\max}/\mu) \cdot \log(1/\epsilon)\right),$$

which is much lower than that of existing "batch" RBCD methods, $\mathcal{O}\left(nkL_{\max}/\mu \cdot \log(1/\epsilon)\right)$.

**Remark 4.4** (Connection to the MRBCD-III method)**.** *There still exists a gap between the theory and empirical success of the active set strategy and its in existing literature, even for the "batch" RBCD methods. When incorporating the active set strategy to the RBCD-style methods, we have known that the empirical performance can be greatly boosted. How to exactly characterize the theoretical speed up is still largely unknown. Therefore Theorem 4.2 and 4.3 can only serve as an imprecise characterization of the MRBCD-III method. A rough understanding is that if the solution has at most $q$ nonzero entries throughout all iterations, then the MRBCD-III method should have an approximate overall iteration complexity*

$$\mathcal{O}\left((nk + qT_{\max}/\mu) \cdot \log(1/\epsilon)\right).$$

## 4.2 Nonstrongly Convex Functions

When $\mathcal{P}(\boldsymbol{\theta})$ is not strongly convex, we can adopt a perturbation approach. Instead of solving (1.1), we consider the minimization problem as follows,

$$\vec{\boldsymbol{\theta}} = \operatorname*{argmin}_{\boldsymbol{\theta} \in \mathbb{R}^d} \mathcal{F}(\boldsymbol{\theta}) + \gamma||\boldsymbol{\theta}^{(0)} - \boldsymbol{\theta}||^2 + \mathcal{R}(\boldsymbol{\theta}), \tag{4.1}$$

where $\gamma$ is some positive perturbation parameter, and $\boldsymbol{\theta}^{(0)}$ is the initial value. If we consider $\widetilde{\mathcal{F}}(\boldsymbol{\theta}) = \mathcal{F}(\boldsymbol{\theta}) + \gamma||\boldsymbol{\theta}^{(0)} - \boldsymbol{\theta}||^2$ in (4.1) as the smooth empirical risk function, then $\widetilde{\mathcal{F}}(\boldsymbol{\theta})$ is a strongly convex function. Thus Corollary 4.3 can be applied to (4.1): When $\mathcal{B}$, $m$, $\eta$, and $\rho$ are suitably chosen, given

$$s \geq 3\log([\mathcal{P}(\boldsymbol{\theta}^{(0)}) - \mathcal{P}(\vec{\boldsymbol{\theta}}) - \gamma||\boldsymbol{\theta}^{(0)} - \vec{\boldsymbol{\theta}}||^2]/\rho) + 3\log(2/\epsilon),$$

we have $\mathcal{P}(\widetilde{\boldsymbol{\theta}}^{(s)}) - \mathcal{P}(\vec{\boldsymbol{\theta}}) - \gamma||\boldsymbol{\theta}^{(0)} - \vec{\boldsymbol{\theta}}||^2 \le \epsilon/2$ with at least probability $1 - \rho$. We then have

$$\mathcal{P}(\widetilde{\boldsymbol{\theta}}^{(s)}) - \mathcal{P}(\widehat{\boldsymbol{\theta}}) \le \mathcal{P}(\widetilde{\boldsymbol{\theta}}^{(s)}) - \mathcal{P}(\widehat{\boldsymbol{\theta}}) - \gamma||\boldsymbol{\theta}^{(0)} - \widehat{\boldsymbol{\theta}}||^2 + \gamma||\boldsymbol{\theta}^{(0)} - \widehat{\boldsymbol{\theta}}||^2$$
$$\le \mathcal{P}(\widetilde{\boldsymbol{\theta}}^{(s)}) - \mathcal{P}(\vec{\boldsymbol{\theta}}) - \gamma||\boldsymbol{\theta}^{(0)} - \vec{\boldsymbol{\theta}}||^2 + \gamma||\boldsymbol{\theta}^{(0)} - \widehat{\boldsymbol{\theta}}||^2 \le \epsilon/2 + \gamma||\boldsymbol{\theta}^{(0)} - \widehat{\boldsymbol{\theta}}||^2.$$

where the second inequality comes from the fact that $\mathcal{P}(\vec{\boldsymbol{\theta}}) + \gamma||\boldsymbol{\theta}^{(0)} - \vec{\boldsymbol{\theta}}||^2 \le \mathcal{P}(\boldsymbol{\theta}) + \gamma||\boldsymbol{\theta}^{(0)} - \widehat{\boldsymbol{\theta}}||^2$, because $\vec{\boldsymbol{\theta}}$ is the minimizer to (4.1). If we choose $\gamma = \epsilon/||\boldsymbol{\theta}^{(0)} - \widehat{\boldsymbol{\theta}}||^2$, we have $\mathcal{P}(\widetilde{\boldsymbol{\theta}}^{(s)}) - \mathcal{P}(\widehat{\boldsymbol{\theta}}) \le \epsilon$. Since $\gamma$ depends on the desired accuracy $\epsilon$, the number of estimated partial gradients also depends on $\epsilon$. Thus if we consider $||\boldsymbol{\theta}^{(0)} - \widehat{\boldsymbol{\theta}}||^2$ as a constant, then the overall iteration complexity of the perturbation approach becomes $\mathcal{O}\left((nk + kT_{\max}/\epsilon) \cdot \log(1/\epsilon)\right)$.

## 5    Numerical Simulations

The first sparse learning problem of our interest is Lasso, which solves

$$\widehat{\boldsymbol{\theta}} = \operatorname*{argmin}_{\boldsymbol{\theta} \in \mathbb{R}^d} \frac{1}{n} \sum_{i=1}^{n} f_i(\boldsymbol{\theta}) + \lambda||\boldsymbol{\theta}||_1 \quad \text{with } f_i = \frac{1}{2}(y_i - \boldsymbol{x}_i^T \boldsymbol{\theta})^2. \tag{5.1}$$

We set $n = 2000$ and $d = 1000$, and all covariate vectors $\boldsymbol{x}_i$'s are independently sampled from a 1000-dimensional Gaussian distribution with mean $\mathbf{0}$ and covariance matrix $\boldsymbol{\Sigma}$, where $\boldsymbol{\Sigma}_{jj} = 1$ and $\boldsymbol{\Sigma}_{jk} = 0.5$ for all $k \ne j$. The first 50 entries of the regression coefficient vector $\boldsymbol{\theta}$ are independently samples from a uniform distribution over support $(-2, -1) \bigcup (+1, +2)$. The responses $y_i$'s are generated by the linear model $y_i = \boldsymbol{x}_i^T \boldsymbol{\theta} + \epsilon_i$, where all $\epsilon_i$'s are independently sampled from a standard Gaussian distribution $N(0, 1)$.

We choose $\lambda = \sqrt{\log d/n}$, and compare the proposed MRBCD-I and MRBCD-II methods with the "batch" proximal gradient (BPG) method [11], the stochastic proximal variance reduced gradient method (SPVRG) [21], and the "batch" randomized block coordinate descent (BRBCD) method [12]. We set $k = 100$. All blocks are of the same size (10 coordinates). For BPG, the step size is $1/T$, where $T$ is the largest singular value of $\frac{1}{n}\sum_{i=1}^{n} \boldsymbol{x}_i \boldsymbol{x}_i^T$. For BRBCD, the step size as $1/L$, where $L$ is the maximum over the largest singular values of $\frac{1}{n}\sum_{i=1}^{n}[\boldsymbol{x}_i]_{G_j}$ of all blocks. For SPVRG, we choose $m = n$, and the step size is $1/(4T)$. For MRBCD-I, the step size is $1/(L\lceil t/8000 \rceil)$, where $t$ is the iteration index. For MRBCD-II, we choose $m = n$, and the step size is $1/(4L)$. Note that the step size and number of iterations $m$ within each inner loop for MRBCD-II and SPVRG are tuned over a refined grid such that the best computational performance is obtained.

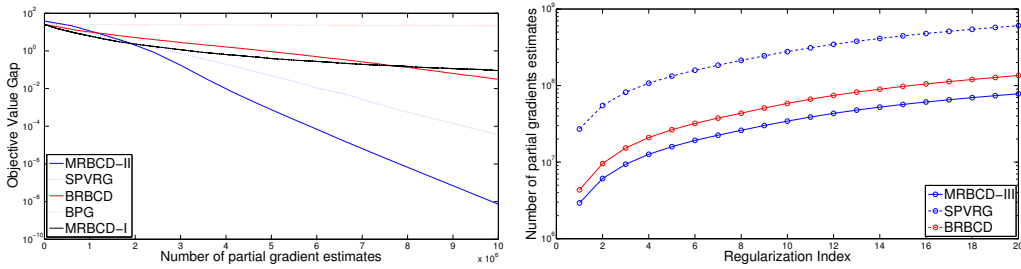

(a) Comparison between different methods for a single regularization parameter.   (b) Comparison between different methods for a sequence of regularization parameters.

Figure 5.1: [a] The vertical axis corresponds to objective value gaps $\mathcal{P}(\boldsymbol{\theta}) - \mathcal{P}(\widehat{\boldsymbol{\theta}})$ in log scale. The horizontal axis corresponds to numbers of partial gradient estimates. [b] The horizontal axis corresponds to indices of regularization parameters. The vertical axis corresponds to numbers of partial gradient estimates in log scale. We see that MRBCD attains the best performance among all methods for both settings

We evaluate the computational performance by the number of estimated partial gradients, and the results averaged over 100 replications are presented in Figure 5.1 [a]. As can be seen, MRBCD-II outperforms SPVRG, and attains the best performance among all methods. The BRBCD and BPG

perform worse than MRBCD-II and SPVRG due to high computational complexity within each iteration. MRBCD-I is actually the fastest among all methods at the first few iterations, and then falls behind SPG and SPVRG due to its sublinear rate of convergence.

We then compare the proposed MRBCD-III method with SPVRG and BRBCD for a sequence of regularization parameters. The sequence contains 21 regularization parameters $\{\lambda_0, ..., \lambda_{20}\}$. We set $\lambda_0 = ||\frac{1}{n}\sum_i y_i \boldsymbol{x}_i||_\infty$, which yields a null solution (all entries are zero), and $\lambda_{20} = \sqrt{\log d/n}$. For $K = 1, ..., 19$, we set $\lambda_K = \alpha \lambda_{K-1}$, where $\alpha = (\lambda_{20}/\lambda_0)^{1/20}$. When solving (5.1) with respect to $\lambda_K$, we use the output solution for $\lambda_{K-1}$ as the initial solution. The above setting is often referred to the warm start scheme in existing literature, and it is very natural to sparse learning problems, since we always need to tune the regularization parameter $\lambda$ to secure good finite sample performance. For each regularization parameter, the algorithm terminates the iteration when the approximate KKT condition is satisfied with $\epsilon = 10^{-10}$.

The results over 50 replications are presented in Figure 5.1 [b]. As can be seen, MRBCD-III outperforms SPVRG and BRBCD, and attains the best performance among all methods. Since BRBCD is also combined with the active set strategy, it attains better performance than SPVRG. See more detailed results in Table E.1 in Appendix E

## 6  Real Data Example

The second sparse learning problem is the elastic-net regularized logistic regression, which solves

$$\widehat{\boldsymbol{\theta}} = \operatorname*{argmin}_{\boldsymbol{\theta} \in \mathbb{R}^d} \frac{1}{n}\sum_{i=1}^n f_i(\boldsymbol{\theta}) + \lambda_1 ||\boldsymbol{\theta}||_1 \quad \text{with} \quad f_i = \log(1 + \exp(-y_i \boldsymbol{x}_i^T \boldsymbol{\theta})) + \frac{\lambda_2}{2}||\boldsymbol{\theta}||^2.$$

We adopt the rcv1 dataset with $n = 20242$ and $d = 47236$. We set $k = 200$, and each block contains approximately 237 coordinates.

We choose $\lambda_2 = 10^{-4}$, and $\lambda_1 = 10^{-4}$, and compare MRBCD-II with SPVRG and BRBCD. For BRBCD, the step size as $1/(4L)$, where $L$ is the maximum of the largest singular values of $\frac{1}{n}\sum_{i=1}^n [\boldsymbol{x}_i]_{G_j}$ over all blocks for BRBCD. For SPVRG, $m = n$ and the step size is $1/(16T)$, where $T$ is the largest singular value of $1/\frac{1}{n}\sum_{i=1}^n \boldsymbol{x}_i \boldsymbol{x}_i^T$. For MRBCD-II, $m = n$ and the step size is $1/(16T)$. For BRBCD, the step size as $1/(4L)$, where $L = \frac{1}{n}\max_j \sum_{i=1}^n [\boldsymbol{x}_i]_j^2$ for BRBCD. Note that the step size and number of iterations $m$ within each inner loop for MRBCD-II and SPVRG are tuned over a refined grid such that the best computational performance is obtained.

The results averaged over 30 replications are presented in Figure F.1 [a] of Appendix F. As can be seen, MRBCD-II outperforms SPVRG, and attains the best performance among all methods. The BRBCD performs worse than MRBCD-II and SPVRG due to high computational complexity within each iteration.

We then compare the proposed MRBCD-III method with SPVRG and BRBCD for a sequence of regularization parameters. The sequence contains 11 regularization parameters $\{\lambda_0, ..., \lambda_{10}\}$. We set $\lambda_0 = ||\frac{1}{n}\sum_i \nabla f_i(\boldsymbol{0})||_\infty$, which yields a null solution (all entries are zero), and $\lambda_{10} = 1e-4$. For $K = 1, ..., 9$, we set $\lambda_K = \alpha \lambda_{K-1}$, where $\alpha = (\lambda_{10}/\lambda_0)^{1/10}$. For each regularization parameter, we set $\epsilon = 10^{-7}$ for the approximate KKT condition.

The results over 30 replications are presented in Figure F.1 [b] of Appendix F. As can be seen, MRBCD-III outperforms SPVRG and BRBCD, and attains the best performance among all methods. Since BRBCD is also combined with the active set strategy, it attains better performance than SPVRG.

**Acknowledgements** This work is partially supported by the grants NSF IIS1408910, NSF IIS1332109, NIH R01MH102339, NIH R01GM083084, and NIH R01HG06841. Yu is supported by China Scholarship Council and by NSFC 61173073.

# References

[1] Amir Beck and Marc Teboulle. A fast iterative shrinkage-thresholding algorithm for linear inverse problems. *SIAM Journal on Imaging Sciences*, 2(1):183–202, 2009.

[2] S. Boyd and L. Vandenberghe. *Convex Optimization*. Cambridge University Press, 2009.

[3] John Duchi and Yoram Singer. Efficient online and batch learning using forward backward splitting. *The Journal of Machine Learning Research*, 10:2899–2934, 2009.

[4] Jerome Friedman, Trevor Hastie, Holger Höfling, and Robert Tibshirani. Pathwise coordinate optimization. *The Annals of Applied Statistics*, 1(2):302–332, 2007.

[5] Rie Johnson and Tong Zhang. Accelerating stochastic gradient descent using predictive variance reduction. In *Advances in Neural Information Processing Systems*, pages 315–323, 2013.

[6] Jakub Konečnỳ and Peter Richtárik. Semi-stochastic gradient descent methods. *arXiv preprint arXiv:1312.1666*, 2013.

[7] John Langford, Lihong Li, and Tong Zhang. Sparse online learning via truncated gradient. *Journal of Machine Learning Research*, 10(777-801):65, 2009.

[8] Han Liu, Mark Palatucci, and Jian Zhang. Blockwise coordinate descent procedures for the multi-task lasso, with applications to neural semantic basis discovery. In *Proceedings of the 26th Annual International Conference on Machine Learning*, pages 649–656, 2009.

[9] L. Meier, S. Van De Geer, and P Bühlmann. The group lasso for logistic regression. *Journal of the Royal Statistical Society: Series B*, 70(1):53–71, 2008.

[10] Sahand N Negahban, Pradeep Ravikumar, Martin J Wainwright, and Bin Yu. A unified framework for high-dimensional analysis of $m$-estimators with decomposable regularizers. *Statistical Science*, 27(4):538–557, 2012.

[11] Yu Nesterov. Gradient methods for minimizing composite objective function. Technical report, Université catholique de Louvain, Center for Operations Research and Econometrics (CORE), 2007.

[12] Peter Richtárik and Martin Takáč. Iteration complexity of randomized block-coordinate descent methods for minimizing a composite function. *arXiv preprint arXiv:1107.2848*, 2011.

[13] Peter Richtárik and Martin Takáč. Iteration complexity of randomized block-coordinate descent methods for minimizing a composite function. *Mathematical Programming*, pages 1–38, 2012.

[14] Nicolas L Roux, Mark Schmidt, and Francis R Bach. A stochastic gradient method with an exponential convergence _rate for finite training sets. In *Advances in Neural Information Processing Systems*, pages 2672–2680, 2012.

[15] Shai Shalev-Shwartz and Ambuj Tewari. Stochastic methods for $\ell_1$-regularized loss minimization. *The Journal of Machine Learning Research*, 12:1865–1892, 2011.

[16] Suvrit Sra, Sebastian Nowozin, and Stephen J Wright. *Optimization for machine learning*. Mit Press, 2012.

[17] R. Tibshirani. Regression shrinkage and selection via the lasso. *Journal of the Royal Statistical Society, Series B*, 58(1):267–288, 1996.

[18] Huahua Wang and Arindam Banerjee. Randomized block coordinate descent for online and stochastic optimization. *CoRR*, abs/1407.0107, 2014.

[19] Li Wang, Ji Zhu, and Hui Zou. The doubly regularized support vector machine. *Statistica Sinica*, 16(2):589, 2006.

[20] Tong Tong Wu and Kenneth Lange. Coordinate descent algorithms for lasso penalized regression. *The Annals of Applied Statistics*, 2:224–244, 2008.

[21] Lin Xiao and Tong Zhang. A proximal stochastic gradient method with progressive variance reduction. *arXiv preprint arXiv:1403.4699*, 2014.

[22] Ming Yuan and Yi Lin. Model selection and estimation in the gaussian graphical model. *Biometrika*, 94(1):19–35, 2007.

[23] Ji Zhu, Saharon Rosset, Trevor Hastie, and Robert Tibshirani. 1-norm support vector machines. In *NIPS*, volume 15, pages 49–56, 2003.

[24] Hui Zou and Trevor Hastie. Regularization and variable selection via the elastic net. *Journal of the Royal Statistical Society: Series B (Statistical Methodology)*, 67(2):301–320, 2005.

